# A Differential Semantics for Jointree Algorithms

**James D. Park and Adnan Darwiche**
Computer Science Department
University of California, Los Angeles, CA 90095
{jd,darwiche}@cs.ucla.edu

## Abstract

A new approach to inference in belief networks has been recently proposed, which is based on an algebraic representation of belief networks using multi–linear functions. According to this approach, the key computational question is that of representing multi–linear functions compactly, since inference reduces to a simple process of evaluating and differentiating such functions. We show here that mainstream inference algorithms based on jointrees are a special case of this approach in a very precise sense. We use this result to prove new properties of jointree algorithms, and then discuss some of its practical and theoretical implications.

## 1   Introduction

It was recently shown that the probability distribution of a belief network can be represented using a multi–linear function, and that most probabilistic queries of interest can be retrieved directly from the partial derivatives of this function [2]. Although the multi–linear function has an exponential number of terms, it can be represented using a small arithmetic circuit in certain situations [3].[1]  Once a belief network is represented as an arithmetic circuit, probabilistic inference is then performed by evaluating and differentiating the circuit, using a very simple procedure which resembles back–propagation in neural networks.

We show in this paper that mainstream inference algorithms based on jointrees [14, 8] are a special-case of the arithmetic–circuit approach proposed in [2]. Specifically, we show that each jointree is an implicit representation of an arithmetic circuit; that the inward–pass in jointree propagation evaluates this circuit; and that the outward–pass differentiates the circuit. Using these results, we prove new useful properties of jointree propagation. We also suggest a new interpretation of the process of factoring graphical models into jointrees, as a process of factoring exponentially–sized multi–linear functions into arithmetic circuits of smaller size.

| A | B | |
|------|-------|----------------------|
| true | true | $\theta_{b\mid a} = .2$ |
| true | false | $\theta_{\bar{b}\mid a} = .8$ |
| false | true | $\theta_{b\mid \bar{a}} = .7$ |
| false | false | $\theta_{\bar{b}\mid \bar{a}} = .3$ |

| A | |
|-------|----------------|
| true | $\theta_a = .6$ |
| false | $\theta_{\bar{a}} = .4$ |

| A | C | |
|-------|-------|-------------------------|
| true | true | $\theta_{c\mid a} = .8$ |
| true | false | $\theta_{\bar{c}\mid a} = .2$ |
| false | true | $\theta_{c\mid \bar{a}} = .15$ |
| false | false | $\theta_{\bar{c}\mid \bar{a}} = .85$ |

Figure 1: The CPTs of belief network $B \leftarrow A \rightarrow C$.

This paper is structured as follows. Sections 2 and 3 are dedicated to a review of inference approaches based on arithmetic circuits and jointrees. Section 4 shows that the jointree approach is a special case of the arithmetic–circuit approach, and discusses some practical implications of this finding. Finally, Section 5 closes with a new perspective on factoring graphical models. Proofs of all theorems can be found in the long version of this paper [11].

## 2 Belief networks as multi–linear functions

A belief network is a factored representation of a probability distribution. It consists of two parts: a directed acyclic graph (DAG) and a set of conditional probability tables (CPTs). For each node $X$ and its parents $\mathbf{U}$, we have a CPT that specifies the distribution of $X$ given each instantiation $\mathbf{u}$ of the parents; see Figure 1.[2]

A belief network is a *representational* factorization of a probability distribution, not a *computational* one. That is, although the network compactly represents the distribution, it needs to be processed further if one is to obtain answers to arbitrary probabilistic queries. Mainstream algorithms for inference in belief networks operate on the network to generate a *computational* factorization, allowing one to answer queries in time which is linear in the factorization size. A most influential computational factorization of belief networks is the *jointree* [14, 8, 6]. Standard jointree factorizations are structure–based: their size depend only on the network topology and is invariant to local CPT structure. This observation has triggered much research for alternative, finer–grained factorizations, since real-world networks can exhibit significant local structure that leads to smaller factorizations if exploited.

We discuss next one of the latest proposals in this direction, which calls for using *arithmetic circuits* as a computational factorization of belief networks [2]. This proposal is based on viewing each belief network as a multi–linear function, which can be represented compactly using an arithmetic circuit. The multi–linear function itself contains two types of variables. First, *evidence indicators,* where for each variable $X$ in the network , we have a variable $\lambda_x$ for each value $x$ of $X$. Second, *network parameters,* where for each variable $X$ and its parents $\mathbf{U}$ in the network, we have a variable $\theta_{x\mid \mathbf{u}}$ for each value $x$ of $X$ and instantiation $\mathbf{u}$ of $\mathbf{U}$.

The multi–linear function has a term for each instantiation of the network variables, which is constructed by multiplying all evidence indicators and network parameters that are consistent with that instantiation. For example, the multi–linear function of the network in Figure 1 has eight terms corresponding to the eight instantiations of variables $A, B, C$: $f = \lambda_a \lambda_b \lambda_c \theta_a \theta_{b\mid a} \theta_{c\mid a} + \lambda_a \lambda_b \lambda_{\bar{c}} \theta_a \theta_{b\mid a} \theta_{\bar{c}\mid a} + \ldots + \lambda_{\bar{a}} \lambda_{\bar{b}} \lambda_{\bar{c}} \theta_{\bar{a}} \theta_{\bar{b}\mid \bar{a}} \theta_{\bar{c}\mid \bar{a}}$. We will often refer to such a multi–linear function as the *network polynomial.*

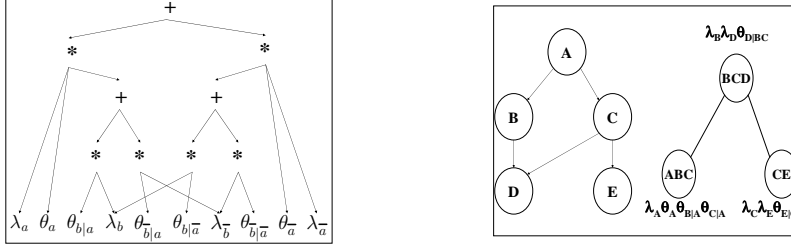

Figure 2: On the left: An arithmetic circuit which computes the function $\lambda_a\lambda_b\theta_a\theta_{b|a} + \lambda_a\lambda_{\bar{b}}\theta_a\theta_{\bar{b}|a} + \lambda_{\bar{a}}\lambda_b\theta_{\bar{a}}\theta_{b|\bar{a}} + \lambda_{\bar{a}}\lambda_{\bar{b}}\theta_{\bar{a}}\theta_{\bar{b}|\bar{a}}$. The circuit is a DAG, where leaf nodes represent function variables and internal nodes represent arithmetic operations. On the right: A belief network structure and its corresponding jointree.

Given the network polynomial $f$, we can answer any query with respect to the belief network. Specifically, let $\mathbf{e}$ be an instantiation of some network variables, and suppose we want to compute the probability of $\mathbf{e}$. We can do this by simply evaluating the polynomial $f$ while setting each evidence indicator $\lambda_x$ to 1 if $x$ is consistent with $\mathbf{e}$, and to 0 otherwise. For the network in Figure 1, we can compute the probability of evidence $\mathbf{e} = b\bar{c}$ by evaluating its polynomial above under $\lambda_a = 1, \lambda_{\bar{a}} = 1, \lambda_b = 1, \lambda_{\bar{b}} = 0$ and $\lambda_c = 0, \lambda_{\bar{c}} = 1$. This leads to $\theta_a\theta_{b|a}\theta_{\bar{c}|a} + \theta_{\bar{a}}\theta_{b|\bar{a}}\theta_{\bar{c}|\bar{a}}$, which equals the probability of $b, \bar{c}$ in this case. We use $f(\mathbf{e})$ to denote the result of evaluating the polynomial $f$ under evidence $\mathbf{e}$ as given above.

This algebraic representation of belief networks is attractive as it allows us to obtain answers to a large number of probabilistic queries directly from the derivatives of the network polynomial [2]. For example, the posterior marginal $Pr(x|\mathbf{e})$ for a variable $X \notin \mathbf{E}$ equals $\frac{1}{f(\mathbf{e})}\frac{\partial f(\mathbf{e})}{\partial \lambda_x}$, where $\frac{\partial f(\mathbf{e})}{\partial \lambda_x}$ is the partial derivative of $f$ wrt $\lambda_x$ evaluated at $\mathbf{e}$. Second, the probability of evidence $\mathbf{e}$ after having retracted the value of some variable $X$ from $\mathbf{e}$, $Pr(\mathbf{e} - X)$, equals $\sum_x \frac{\partial f(\mathbf{e})}{\partial \lambda_x}$. Third, the posterior marginal $Pr(x, \mathbf{u}|\mathbf{e})$ for a variable $X$ and its parents $\mathbf{U}$ equals $\frac{\theta_{x|\mathbf{u}}}{f(\mathbf{e})}\frac{\partial f(\mathbf{e})}{\partial \theta_{x|\mathbf{u}}}$.

The network polynomial has an exponential number of terms, yet one can represent it compactly in certain cases using an arithmetic circuit; see Figure 2. The (first) partial derivatives of an arithmetic circuit can all be computed simultaneously in time linear in the circuit size [2, 12]. The procedure resembles the back–propagation algorithm for neural networks as it evaluates the circuit in a single upward–pass, and then differentiates it through a single downward–pass.

The main computational question is then that of generating the smallest arithmetic circuit that computes the network polynomial. A structure–based approach for this has been given in [2], which is guaranteed to generate a circuit whose size is bounded by $O(n\exp(w))$, where $n$ is the number of nodes in the network and $w$ is its treewidth. A more recent approach, however, which exploits local structure has been presented in [3] and was shown experimentally to generate small arithmetic circuits for networks whose treewidth is up to 60. As we show in the rest of this paper, the process of factoring a belief network into a jointree is yet another method for generating an arithmetic circuit for the network. Specifically, we show that the jointree structure is an implicit representation of such a circuit, and that jointree propagation corresponds to circuit evaluation and differentiation. Moreover, the difference between Shenoy–Shafer and Hugin propagation turns out to be a difference in the numeric scheme used for circuit differentiation [11].

# 3  Jointree Algorithms

We now review jointree algorithms, which are quite influential in graphical models. Let $\mathcal{B}$ be a belief network. A jointree for $\mathcal{B}$ is a pair $(\mathcal{T}, \mathcal{L})$, where $\mathcal{T}$ is a tree and $\mathcal{L}$ is a function that assigns labels to nodes in $\mathcal{T}$. A jointree must satisfy three properties: (1) each label $\mathcal{L}(i)$ is a set of variables in the belief network; (2) each network variable $X$ and its parents $\mathbf{U}$ (a family) must appear together in some label $\mathcal{L}(i)$; (3) if a variable appears in the labels of $i$ and $j$, it must also appear in the label of each node $k$ on the path connecting them. The label of edge $ij$ in $\mathcal{T}$ is defined as $\mathcal{L}(i) \cap \mathcal{L}(j)$. We will refer to the nodes of a jointree (and sometimes their labels) as *clusters.* We will also refer to the edges of a jointree (and sometimes their labels) as *separators.* Figure 2 depicts a belief network and one of its jointrees.

Jointree algorithms start by constructing a jointree for a given belief network [14, 8, 6]. They also associate *tables* (also called *potentials*) with clusters and separators.[3] The *conditional probability table* (CPT or CP Table) of each variable $X$ with parents $\mathbf{U}$, denoted $\theta_{X|\mathbf{U}}$, is assigned to a cluster that contains $X$ and $\mathbf{U}$. In addition, an *evidence table* over variable $X$, denoted $\lambda_X$, is assigned to a cluster that contains $X$. Figure 2 depicts the assignments of evidence and CP tables to clusters. Evidence $\mathbf{e}$ is entered into a jointree by initializing evidence tables as follows: we set $\lambda_X(x)$ to 1 if $x$ is consistent with evidence $\mathbf{e}$, and we set $\lambda_X(x)$ to 0 otherwise.

Given some evidence $\mathbf{e}$, a jointree algorithm propagates messages between clusters. After passing two message per edge in the jointree, one can compute the marginals $Pr(\mathbf{C}, \mathbf{e})$ for every cluster $\mathbf{C}$. There are two main methods for propagating messages in a jointree: the Shenoy–Shafer architecture [14] and the Hugin architecture [8].

Shenoy–Shafer propagation proceeds as follows [14]. First, evidence $\mathbf{e}$ is then entered into the jointree. A cluster is then selected as the root and message propagation proceeds in two phases, inward and outward. In the inward phase, messages are passed toward the root. In the outward phase, messages are passed away from the root. Cluster $i$ sends a message to cluster $j$ only when it has received messages from all its other neighbors $k$. A message from cluster $i$ to cluster $j$ is a table $M_{ij}$ defined as follows: $M_{ij} = \sum_{\mathbf{C} \setminus \mathbf{S}} \phi_i \prod_{k \neq j} M_{ki}$, where $\mathbf{C}$ are the variables of cluster $i$, $\mathbf{S}$ are the variables of separator $ij$, and $\phi_i$ is the multiplication of all evidence and CP tables assigned to cluster $i$. Once message propagation is finished, we have $Pr(\mathbf{C}, \mathbf{e}) = \phi_i \prod_k M_{ki}$, where $\mathbf{C}$ are the variables of cluster $i$.

Hugin propagation proceeds similarly to Shenoy–Shafer by entering evidence; selecting a cluster as root; and propagating messages in two phases, inward and outward [8]. The Hugin method, however, differs in some major ways. It maintains a table $\Phi_{ij}$ with each separator, whose entries are initialized to 1s. It also maintains a table $\Phi_i$ with each cluster $i$, initialized to the multiplication of all CPTs and evidence tables assigned to cluster $i$. Cluster $i$ passes a message to neighboring cluster $j$ only when $i$ has received messages from all its other neighbors $k$. When cluster $i$ is ready to send a message to cluster $j$, it does the following. First, it saves the table of separator $\Phi_{ij}$ into $\Phi_{ij}^{old}$. Second, it computes a new separator table $\Phi_{ij} = \sum_{\mathbf{C} \setminus \mathbf{S}} \Phi_i$, where $\mathbf{C}$ are the variables of cluster $i$ and $\mathbf{S}$ are the variables of separator $ij$. Third, it computes a message to cluster $j$: $M_{ij} = \frac{\Phi_{ij}}{\Phi_{ij}^{old}}$. Finally, it multiplies the computed message into the table of cluster $j$: $\Phi_j = \Phi_j M_{ij}$. After the inward and outward–passes of Hugin propagation are completed, we have: $Pr(\mathbf{C}, \mathbf{e}) = \Phi_i$, where $\mathbf{C}$ are the variables of cluster $i$.

# 4  Jointrees as arithmetic circuits

We now show that every jointree (together with a root cluster and a particular assignment of evidence and CP tables to clusters) corresponds precisely to an arithmetic circuit that computes the network polynomial. We also show that the inward–pass of the Shenoy–Shafer architecture evaluates this circuit, while the outward–pass differentiates it. We show a similar result for the Hugin architecture.

**Definition 1** *Given a root cluster, a particular assignment of evidence and CP tables to clusters, the arithmetic circuit embedded in a jointree is defined as follows:*[4]

*Nodes:* *The circuit includes: an output addition node $f$; an addition node $\mathbf{s}$ for each instantiation of a separator $\mathbf{S}$; a multiplication node $\mathbf{c}$ for each instantiation of a cluster $\mathbf{C}$; an input node $\lambda_x$ for each instantiation $x$ of variable $X$; an input node $\theta_{x|\mathbf{u}}$ for each instantiation $x\mathbf{u}$ of family $X\mathbf{U}$.*

*Edges:* *The children of the output node $f$ are the multiplication nodes generated by the root cluster; the children of an addition node $\mathbf{s}$ are all compatible nodes generated by the child cluster; the children of a multiplication node $\mathbf{c}$ are all compatible nodes generated by child separators, and all compatible input nodes assigned to cluster $\mathbf{C}$.*

Hence, separators contribute addition nodes and clusters contribute multiplication nodes. Moreover, the structure of the jointree dictates how these nodes are connected into a circuit. The arithmetic circuit in Figure 2 is embedded in the jointree $A - AB$, with cluster $A$ as the root, and with tables $\lambda_A, \theta_A$ assigned to cluster $A$ and tables $\lambda_B$ and $\theta_{B|A}$ assigned to cluster $B$.

**Theorem 1** *The circuit embedded in a jointree computes the network polynomial.*

Therefore, by constructing a jointree one is generating a compact representation of the network polynomial in terms of an arithmetic circuit.

We are now ready to state our basic results on the differential semantics of jointree propagation, but we need some notational conventions first. In the following three theorems: $f$ denotes the circuit embedded in a jointree or its (unique) output node; $\mathbf{s}$ denotes a separator instantiation or the addition node generated by that instantiation; and $\mathbf{c}$ denotes a cluster instantiation or the multiplication node generated by that instantiation. Moreover, the value that a circuit node $v$ takes under evidence $\mathbf{e}$ is denoted $v(\mathbf{e})$. Recall that a circuit (or network polynomial) is evaluated under evidence $\mathbf{e}$ by setting each input $\lambda_x$ to 1 if $x$ is consistent with $\mathbf{e}$; and to 0 otherwise. Finally, recall that $\partial f/\partial v$ represents the derivative of the circuit output with respect to node $v$. Our first result relates to Shenoy–Shafer propagation.

**Theorem 2** *The messages produced using Shenoy–Shafer propagation on a jointree under evidence $\mathbf{e}$ have the following semantics. For each inward message $M_{ij}$, we have $M_{ij}(\mathbf{s}) = \mathbf{s}(\mathbf{e})$. For each outward message $M_{ji}$, we have $M_{ji}(\mathbf{s}) = \frac{\partial f(\mathbf{e})}{\partial \mathbf{s}}$.*

Hence, if we interpret separator instantiations as addition nodes in a circuit as given by Definition 1, we get that a message directed towards the jointree root contains the values of these addition nodes, while a message directed outward from the root contains the partial derivatives of the circuit output with respect to these nodes.

Shenoy–Shafer propagation does not compute derivatives with respect to input nodes $\lambda_x$ and $\theta_{x|\mathbf{u}}$, but these can be obtained using local computations as follows.

**Theorem 3** *If evidence table $\lambda_X$ is assigned to cluster $i$ with variables $\mathbf{C}$:*

$$\frac{\partial f(\mathbf{e})}{\partial \lambda_x} = \left[ \sum_{\mathbf{C} \backslash X} \prod_j M_{ji} \prod_{\psi \neq \lambda_X} \psi \right] (x), \tag{1}$$

*where $\psi$ ranges over all evidence and CP tables assigned to cluster $i$. Moreover, if CPT $\theta_{X|\mathbf{U}}$ is assigned to cluster $i$ with variables $\mathbf{C}$:*

$$\frac{\partial f(\mathbf{e})}{\partial \theta_{x|\mathbf{u}}} = \left[ \sum_{\mathbf{C} \backslash X\mathbf{U}} \prod_j M_{ji} \prod_{\psi \neq \theta_{X|\mathbf{U}}} \psi \right] (x\mathbf{u}), \tag{2}$$

*where $\psi$ ranges over all evidence and CP tables assigned to cluster $i$.*

Therefore, even though Shenoy–Shafer propagation does not fully differentiate the embedded circuit, the differentiation process can be completed through local computations after propagation has finished.[5]

We now discuss some applications of the partial derivatives with respect to evidence indicators $\lambda_x$ and network parameters $\theta_{x|\mathbf{u}}$.

**Fast retraction & evidence flipping.** Suppose jointree propagation has been performed using evidence $\mathbf{e}$, which gives us access directly to the probability of $\mathbf{e}$. Suppose now we are interested in the probability of a different evidence $\mathbf{e}'$, which results from changing the value of some variable $X$ in $\mathbf{e}$ to a new value $x$. The probability of $\mathbf{e}'$ in this case is equal to $\frac{\partial f(\mathbf{e})}{\partial \lambda_x}$ [2], which can be obtained as given by Equation 1. The ability to perform this computation efficiently is crucial for algorithms that try to approximate *maximum aposteriori hypothesis* (MAP) using local search [9, 10]. Another application of this derivative is in computing the probability of evidence $\mathbf{e}'$, which results from retracting the value of some variable $X$ from $\mathbf{e}$: $Pr(\mathbf{e}') = \sum_x \frac{\partial f(\mathbf{e})}{\partial \lambda_x}$. This computation is key to analyzing evidence conflict, as it allows us to determine the extent to which one piece of evidence is contradicted by the remaining pieces.

**Sensitivity analysis & parameter learning.** The derivative $\frac{\partial Pr(\mathbf{e})}{\partial \theta_{x|\mathbf{u}}}$ is essential for sensitivity analysis—it is the basis for an efficient approach that identifies minimal network parameters changes that are necessary to satisfy constraints on probabilistic queries [1]. This derivative is also crucial for gradient ascent approaches for learning network parameters as it is required to compute the gradient

**Theorem 4** *Cluster tables, separator tables and messages produced using Hugin propagation under evidence $\mathbf{e}$ have the following semantics: For table $\Phi_i$ of cluster $i$ with variables $\mathbf{C}$: $\Phi_i(\mathbf{c}) = \mathbf{c}(\mathbf{e}) \frac{\partial f(\mathbf{e})}{\partial \mathbf{c}}$. For table $\Phi_{ij}$ of separator $ij$ with variables $\mathbf{S}$: $\Phi_{ij}(\mathbf{s}) = \mathbf{s}(\mathbf{e}) \frac{\partial f(\mathbf{e})}{\partial \mathbf{s}}$. For each inward message $M_{ij}$, we have $M_{ij}(\mathbf{s}) = \mathbf{s}(\mathbf{e})$. For each outward message $M_{ji}$, we have $M_{ji}(\mathbf{s}) = \frac{\partial f(\mathbf{e})}{\partial \mathbf{s}}$ if $\mathbf{s}(\mathbf{e}) \neq 0$.*

Again, Hugin propagation does not compute derivatives with respect to input nodes $\lambda_x$ and $\theta_{x|\mathbf{u}}$. Even for addition and multiplication nodes, it only retains derivatives multiplied by values. Hence, if we want to recover the derivative with respect to, say, multiplication node $\mathbf{c}$, we must know the value of this node and it must be different than zero. In such a case, we have $\partial f(\mathbf{e})/\partial \mathbf{c} = \Phi_i(\mathbf{c})/\mathbf{c}(\mathbf{e})$, where $\Phi_i$ is the table associated with the cluster $i$ that generates node $\mathbf{c}$. One can also compute the quantity $v \, \partial f/\partial v$ for input nodes using equations similar to those in Theorem 3. But such quantities will be useful for obtaining derivatives only if the values of such input nodes are not zero. Hence, Shenoy–Shafer propagation is more informative than Hugin propagation as far as the computation of derivatives is concerned.

used for deciding moves in the search space [13]. This derivative equals $\frac{\partial f(\mathbf{e})}{\partial \theta_{x|\mathbf{u}}}$, and can be obtained as given by Equation 2. The only other method we are aware of to compute this derivative (beyond the one in [2]) is the one using the identity $\partial Pr(\mathbf{e})/\partial \theta_{x|\mathbf{u}} = Pr(x, \mathbf{u}, \mathbf{e})/\theta_{x|\mathbf{u}}$, which requires $\theta_{x|\mathbf{u}} \neq 0$ [13]. Hence, our results seem to suggest the first general approach for computing this derivative using standard jointree propagation.

**Bounding rounding errors.** Jointree propagation gives exact results only when infinite precision arithmetic is used. In practice, however, finite precision floating–point arithmetic is typically used, in which case the differential semantics of jointree propagation can be used to bound the rounding error in the computed probability of evidence. See the full paper [11] for details on computing this bound.

## 5    A new perspective on factoring graphical models

We have shown in this paper that each jointree can be viewed as an implicit representation of an arithmetic circuit which computes the network polynomial, and that jointree propagation corresponds to an evaluation and differentiation of the circuit. These results have been useful in unifying the circuit approach presented in [2] with jointree approaches, and in uncovering more properties of jointree propagation.

Another outcome of these results relates to the level at which it is useful to phrase the problem of factoring graphical probabilistic models. Specifically, the perspective we are promoting here is that probability distributions defined by graphical models should be viewed as multi–linear functions, and the construction of jointrees should be viewed as a process of constructing arithmetic circuits that compute these functions. That is, the fundamental object being factored is a multi–linear function, and the fundamental result of the factorization is an arithmetic circuit. A graphical model is a useful abstraction of the multi–linear function, and a jointree is a useful structure for embedding the arithmetic circuit.

This view of factoring is useful since it allows us to cast the factoring problem in more refined terms, which puts us in a better position to exploit the local structure of graphical models in the factorization process. Note that the topology of a graphical model defines the form of the multi–linear function, while the model's local structure (as exhibited in its CPTs) constrains the values of variables appearing in the function. One can factor a multi–linear function without knowledge of such constraints, but the resulting factorizations will not be optimal. For a dramatic example, consider a fully connected network with variables $X_1, \ldots, X_n$, where all parameters are equal to $\frac{1}{2}$. Any jointree for the network will have a cluster of size $n$, leading to $O(\exp(n))$ complexity. There is, however, a circuit of $O(n)$ size here, since the network polynomial can be easily factored as: $f = (\frac{1}{2})^n \prod_{i=1}^{n} (\lambda_{x_i} + \lambda_{\bar{x}_i})$.

Hence, in the presence of local structure, it appears more promising to factor the graphical model into the more refined arithmetic circuit since not every arithmetic circuit can be embedded in a jointree. This promise is made apparent by the results in [3], which we sketch next. First, the multi–linear function of a belief network is "encoded" using a propositional theory, which is expressive enough to capture the form of the multi–linear function in addition to constraints on its variables. The theory is then compiled into a special logical form, known as deterministic decomposable negation normal form. An arithmetic circuit is finally extracted from that form. The method was able to generate relatively small arithmetic circuits for a significant suite of real–world belief networks with treewidths up to 60.

It is worth mentioning here that the above perspective is in harmony with recent

approaches that represent probabilistic models using algebraic decision diagrams (ADDs), citing the promise of ADDs in exploiting local structure [5]. ADDs and related representations, such as edge–valued decision diagrams, are known to be compact representations of multi–linear functions. Moreover, each of these representations can be expanded in linear time into an arithmetic circuit that satisfies some strong properties [4]. Hence, such representations are special cases of arithmetic circuits as well.

We finally note that the relationship between multi–linear functions (polynomials in general) and arithmetic circuits is a classical subject of *algebraic complexity theory* [15]. In this field of complexity, computational problems are expressed as polynomials, and a central question is that of determining the size of the smallest arithmetic circuit that computes a given polynomial, leading to the notion of *circuit complexity.* Using this notion, it is then meaningful to talk about the circuit complexity of a graphical model: the size of the smallest arithmetic circuit that computes the multi–linear function induced by the model.

**Acknowledgment** This work has been partially supported by NSF grant IIS-9988543 and MURI grant N00014-00-1-0617.

## Footnotes

[1]For example, it was shown recently that real–world belief networks with treewidth up to 60 can be compiled into arithmetic circuits with few thousand nodes [3]. Such networks have local structure, and are outside the scope of mainstream algorithms for inference in belief networks whose complexity is exponential in treewidth.

[2]Variables are denoted by upper–case letters ($A$) and their values by lower–case letters ($a$). Sets of variables are denoted by bold–face upper–case letters ($\mathbf{A}$) and their instantiations are denoted by bold–face lower–case letters ($\mathbf{a}$). For a variable $A$ with values *true* and *false*, we use $a$ to denote $A=$ *true* and $\bar{a}$ to denote $A=$ *false*. Finally, for a variable $X$ and its parents $\mathbf{U}$, we use $\theta_{x\mid \mathbf{u}}$ to denote the CPT entry corresponding to $Pr(x \mid \mathbf{u})$.

[3]A table is an array which is indexed by variable instantiations. Specifically, a table $\phi$ over variables $\mathbf{X}$ is indexed by the instantiations $\mathbf{x}$ of $\mathbf{X}$. Its entries $\phi(\mathbf{x})$ are in $[0, 1]$.

[4]Given a root cluster, one can direct the jointree by having arrows point away from the root, which also defines a parent/child relationship between clusters and separators.

[5]Hugin propagation also corresponds to circuit evaluation/differentiation:

# References

[1] H. Chan and A. Darwiche. When do numbers really matter? *JAIR,* 17: 265–287, 2002.

[2] A. Darwiche. A differential approach to inference in Bayesian networks. In *UAI'00,* pages 123–132, 2000. To appear in JACM.

[3] A. Darwiche. A logical approach to factoring belief networks. In *KR'02*, pages 409–420, 2002.

[4] A. Darwiche. On the factorization of multi–linear functions. Technical Report D–128, UCLA, Los Angeles, Ca 90095, 2002.

[5] J. Hoey, R. St-Aubin, A. Hu, and G. Boutilier. SPUDD: Stochastic planning using decision diagrams. In *UAI'99*, pages 279–288, 1999.

[6] C. Huang and A. Darwiche. Inference in belief networks: A procedural guide. *IJAR,* 15(3): 225–263, 1996.

[7] M. Iri. Simultaneous computation of functions, partial derivatives and estimates of rounding error. *Japan J. Appl. Math.*, 1:223–252, 1984.

[8] F. V. Jensen, S.L. Lauritzen, and K.G. Olesen. Bayesian updating in recursive graphical models by local computation. *Comp. Stat. Quart.*, 4:269–282, 1990.

[9] J. Park. MAP complexity results and approximation methods. In *UAI'02*, pages 388–396, 2002.

[10] J. Park and A. Darwiche. Approximating MAP using stochastic local search. In *UAI'01*, pages 403–410, 2001.

[11] J. Park and A. Darwiche. A differential semantics for jointree algorithms. Technical Report D–118, UCLA, Los Angeles, Ca 90095, 2001.

[12] G. Rote. Path problems in graphs. *Computing Suppl.*, 7:155–189, 1990.

[13] S. Russell, J. Binder, D. Koller, and K. Kanazawa. Local learning in probabilistic networks with hidden variables. In *UAI'95*, pages 1146–1152, 1995.

[14] P. P. Shenoy and G. Shafer. Propagating belief functions with local computations. *IEEE Expert*, 1(3):43–52, 1986.

[15] J. von zur Gathen. Algebraic complexity theory. *Ann. Rev. Comp. Sci.*, 3:317–347, 1988.
